# ADAPTIVE SPLINE NETWORKS

Jerome H. Friedman
Department of Statistics and
Stanford Linear Accelerator Center
Stanford University
Stanford, CA 94305

## Abstract

A network based on splines is described. It automatically adapts the number of units, unit parameters, and the architecture of the network for each application.

## 1 INTRODUCTION

In supervised learning one has a system under study that responds to a set of simultaneous input signals $\{x_1 \cdots x_n\}$. The response is characterized by a set of output signals $\{y_1, y_2, \cdots, y_m\}$. The goal is to learn the relationship between the inputs and the outputs. This exercise generally has two purposes: prediction and understanding. With prediction one is given a set of input values and wishes to predict or forecast likely values of the corresponding outputs without having to actually run the system. Sometimes prediction is the only purpose. Often, however, one wishes to use the derived relationship to gain understanding of how the system works. Such knowledge is often useful in its own right, for example in science, or it may be used to help improve the characteristics of the system, as in industrial or engineering applications.

The learning is accomplished by taking training data. One observes the outputs produced by the system in response to varying sets of input values

$$\{y_{1i} \cdots y_{mi} \mid x_{1i} \cdots x_{ni}\}_1^N. \tag{1}$$

These data (1) are then used to train an "artificial" system (usually a computer program) to learn the input/output relationship. The underlying framework or model is usually taken to be

$$y_k = f_k(x_1 \cdots x_n) + \epsilon_k, \quad k = 1, m \tag{2}$$

with $\text{ave}(\epsilon_k \mid x_1 \cdots x_n) = 0$. Here (2) $y_k$ is the $k$th responding output signal, $f_k$ is a single valued deterministic function of an $n$-dimensional argument (inputs) and $\epsilon_k$ is a random (stochastic) component that reflects the fact that (if nonzero) $y_k$ is not completely specified by the observed inputs, but is also responding to other quantities that are neither controlled nor observed. In this framework the learning goal is to use the training data to derive a function $\hat{f}(x_1 \cdots x_n)$ that can serve as a reasonable approximation (estimate) of the true underlying ("target") function $f_k$ (2). The supervised learning problem can in this way be viewed as one of function or surface approximation, usually in high dimensions ($n >> 2$).

## 2    SPLINES

There is an extensive literature on the theory of function approximation (see Cheney [1986] and Chui [1988], and references therein). From this literature spline methods have emerged as being among the most successful (see deBoor [1978] for a nice introduction to spline methods). Loosely speaking, spline functions have the property that they are the smoothest for a given flexibility and vice versa. This is important if one wishes to operate under the least restrictive assumptions concerning $f_k(x_1 \cdots x_n)$ (2), namely, that it is relatively smooth compared to the noise $\epsilon_k$ but is otherwise arbitrary. A spline approximation is characterized by its order $q$ [$q = 1$ (linear), $q = 2$ (quadratic), and $q = 3$ (cubic) are the most popular orders]. The procedure is to first partition the input variable space into a set of disjoint regions. The approximation $\hat{f}(x_1 \cdots x_n)$ is taken to be a separate $n$-dimensional polynomial in each region with maximum degree $q$ in any one variable, constrained so that $\hat{f}$ and all of its derivatives to order $q - 1$ are continuous across all region boundaries. Thus, a particular spline approximation is determined by a choice for $q$, which tends not to be very important, and the particular set of chosen regions, which tends to be crucial. The central problem associated with spline approximations is how to choose a good set of associated regions for the problem at hand.

### 2.1    TENSOR-PRODUCT SPLINES

The most popular method for partitioning the input variable space is by the tensor or outer product of interval sets on each of the $n$ axes. Each input axis is partitioned into $K + 1$ intervals delineated by $K$ points ("knots"). The regions in the $n$-dimensional space are taken to be the $(K + 1)^n$ intersections of all such intervals. Figure 1 illustrates this procedure for $K = 4$ knots on each of two axes producing 25 regions in the corresponding two-dimensional space.

Owing to the regularity of tensor-product representations, the corresponding spline approximation can be represented in a simple form as a basis function expansion. Let $\mathbf{x} = (x_1 \cdots x_n)$. Then

$$\hat{f}(\mathbf{x}) = \sum_{\ell} w_\ell B_\ell(\mathbf{x}) \tag{3}$$

where $\{w_\ell\}$ are the coefficients (weights) for each respective basis function $B_\ell(\mathbf{x})$, and the basis function set $\{B_\ell(\mathbf{x})\}$ is obtained by taking the tensor product of the set of functions

$$\{x_j^i\}_{i=0}^q, \quad \{(x_j - t_{kj})_+^q\}_{k=1}^K \tag{4}$$

over all of the axes, $j = 1, n$. That is, each of the $K + q + 1$ functions on each axis $j$ ($j = 1, n$) is multiplied by all of the functions (4) corresponding to all of the other axes $k$ ($k = 1, n$; $k \neq j$). As a result the total number of basis functions (3) defining the tensor-product spline approximation is

$$(K + q + 1)^n. \tag{5}$$

The functions comprising the second set in (4) are known as the truncated power functions:

$$(x_j - t_{kj})_+^q = \begin{cases} 0 & x_j \leq t_{kj} \\ (x_j - t_{kj})^q & x_j > t_{kj} \end{cases} \tag{6}$$

and there is one for each knot location $t_{kj}$ ($k = 1, K$) on each input axis $j$ ($j = 1, n$).

Although conceptually quite simple, tensor-product splines have severe limitations that preclude their use in high dimensional settings ($n >> 2$). These limitations stem from the exponentially large number of basis functions that are required (5). For cubic splines ($q = 3$) with five inputs ($n = 5$) and only five knots per axis ($K = 5$) 59049 basis functions are required. For $n = 6$ that number is 531441, and for $n = 10$ it is approximately $3.5 \times 10^9$. This poses severe statistical problems in fitting the corresponding number of weights unless the training sample is large compared to these numbers, and computational problems in any case since the computation grows as the number of weights (basis functions) cubed. These are typical manifestations of the so-called "curse-of-dimensionality" (Bellman [1961]) that afflicts nearly all high-dimensional problems.

## 3  ADAPTIVE SPLINES

This section gives a very brief overview of an adaptive strategy that attempts to overcome the limitations of the straightforward application of tensor-product splines, making practical their use in high-dimensional settings. This method, called MARS (multivariate adaptive regression splines), is described in detail in Friedman [1991] along with many examples of its use involving both real and artificially generated data. (A FORTRAN program implementing the method is available from the author.)

The method (conceptually) begins by generating a tensor-product partition of the input variable space using a large number of knots, $K \lesssim N$, on each axis. Here $N$ (1) is the training sample size. This induces a very large $(K + 1)^n$ number of regions. The procedure then uses the training data to select particular unions of these (initially large number of) regions to define a relatively small number of (larger) regions most suitable for the problem at hand.

This strategy is implemented through the basis function representation of spline approximations (3). The idea is to select a relatively small subset of basis functions

$$\{B_m^*(\mathbf{x})\}_0^M \underset{\text{small}}{\subset} \{B_\ell(\mathbf{x})\}_0^{\text{huge}} \tag{7}$$

from the very large set (3) (4) (5) induced by the initial tensor-product partition. The particular subset for a problem at hand is obtained through standard statistical variable subset selection, treating the basis functions as the "variables". At the

first step the best single basis function is chosen. The second step chooses the basis function that works best in conjunction with the first. At the $m$th step, the one that works best with the $m-1$ already selected, is chosen, and so on. The process stops when including additional basis functions fails to improve the approximation.

## 3.1    ADAPTIVE SPLINE NETWORKS

This section describes a network implementation that approximates the adaptive spline strategy described in the previous section. The goal is to synthesize a good set of spline basis functions (7) to approximate a particular system's input/output relationship, using the training data. For the moment, consider only one output $y$; this is generalized later. The basic observation leading to this implementation is that the approximation takes the form of sums of products of very simple functions, namely the truncated power functions (6), each involving a single input variable,

$$B_m^*(\mathbf{x}) = \prod_{k=1}^{K_m} (x_{j(k)} - t_{kj})_+^q, \tag{8}$$

and

$$\hat{f}(\mathbf{x}) = \sum_{m=0}^{M} w_m B_m^*(\mathbf{x}). \tag{9}$$

Here $1 \leq j(k) \leq n$ is an input variable and $1 \leq K_m \leq n$ is the number of factors in the product (interaction level).

The network is comprised of an ordered set of interconnected units. Figure 2 shows a diagram of the interconnections for a (small) network. Figure 3 shows a schematic diagram of each individual unit. Each unit has as its inputs all of the system inputs $x_1 \cdots x_n$ and all of the outputs from the previous units in the network $B_0 \cdots B_M$. It is also characterized by three parameters: $j, \ell, t$. The triangles in Figure 3 represent selectors. The upper triangle selects one of the system inputs, $x_j$; the left triangle selects one of the previous unit outputs, $B_\ell$. These serve as inputs, along with the parameter $t$, to two internal units that each produce an output. The first output is $B_\ell \cdot (x_j - t)_+^q$ and the second is $B_\ell \cdot (t - x_j)_+^q$. The whole unit thereby produces two outputs $B_{M+1}$ and $B_{M+2}$, that are available to serve as inputs to future units. In addition to units of this nature, there is an initial unit ($B_0$) that produces the constant output $B_0 = 1$, that is also available to be selected as an input to all units. The output of the entire network, $\hat{f}$, is a weighted sum (9) of all of the unit outputs (including $B_0 = 1$). This is represented by the bottom trapezoid in Figure 2.

The parameters associated with the network are the number of units $Nu$, the parameters associated with each one

$$\{\ell_i, j_i, t_i\}_1^{Nu}, \tag{10}$$

and the weights in the final adder

$$\{w_k\}_0^{M=2 \cdot Nu}. \tag{11}$$

The goal of training the network is to choose values for these parameters (10) (11) that minimize average future prediction error (squared), that is the squared error on

(test) data not used as part of the training sample. An estimate of this quantity is provided by the generalized cross-validation model selection criterion (Craven and Wahba [1979])

$$GCV = \frac{1}{N} \sum_{i=1}^{N} (y_i - \hat{f}_i)^2 / \left[ 1 - \frac{5 \cdot Nu + 1}{N} \right]^2. \qquad (12)$$

The numerator in (12) is the average squared-error over the training data. The denominator is an (inverse) penalty for adding units. The quantity $(5 \cdot Nu + 1)$ is just the number of adjustable parameters in the network. This GCV criterion (12) has its roots in ordinary (leave-one-out) cross-validation and serves as an approximation to it (see Craven and Wahba [1979]).

The training strategy used is a semi-greedy one. The units are considered in order. For the $m$th unit the weights of all later units are set to zero, that is

$$w_{2m+1} = \cdots = w_{2M_{\max}} = 0$$

where $M_{\max}$ is the maximum number of units in the network. The GCV criterion (12) is then minimized with respect to the parameters of the $m$th unit $(\ell_m, j_m, t_m)$, and the weights associated with all previous units as well as the unit under consideration $\{w_k\}_0^{2m}$, given the parameter values associated with the previous units $\{\ell_i, j_i, t_i\}_1^{m-1}$. This optimization can be done very rapidly, $O(nm^2N)$, using least squares updating formulae (see Friedman [1991]). This process is repeated until $M_{\max}$ units have been added to the network. A post optimization procedure (weight elimination) is then applied to select an optimal subset of weights to be set to zero, so as to minimize the GCV criterion (12). This will (usually) decrease the GCV value since it includes a penalty for increasing the number of nonzero weights

The semi-greedy training strategy has the advantage of being quite fast. The total computation is $O(nNM_{\max}^3)$ where $n$ is the number of system inputs, $N$ is the training sample size, and $M_{\max}$ is the maximum number of units to be included in the network (before weight elimination). On a SUN SPARCstation, small to moderate sized problems train in seconds to minutes, and very large ones in a few hours. A potential disadvantage of this strategy is possible loss of prediction accuracy compared to a more thorough optimization strategy. This tends not to be the case. Experiments with more complete optimization seldom resulted in even moderate improvement. This is because units added later to the network can compensate for the suboptimal settings of parameters introduced earlier.

Figure 4 illustrates a (very small) network that might be realized with the MARS procedure. The number above each unit is the system input that it selected. The letter within each unit represents its knot parameter. The first unit necessarily has as its input the constant $B_0 = 1$. Its first output goes to the final adder but was not selected as an input to any future units. Its second output serves as the selected input to the next two units, but was eliminated from the adder by the final weight elimination, and so on. The final approximation for this network is

$$\hat{f}(\mathbf{x}) = w_0 + w_1(x_3 - s)_+^q + w_2(s - x_3)_+^q (x_7 - t)_+^q$$

$$+ w_3(s - x_3)_+^q (x_2 - u)_+^q + w_4(s - x_3)_+^q (u - x_2)_+^q (x_8 - v)_+^q$$

$$+ w_5(s - x_3)_+^q (u - x_2)_+^q (v - x_8)_+^q.$$

Two possible network topologies that might be realized are of special interest. One is where all units happen to select the constant line $B_0 = 1$ as their unit input. In this case the resulting approximation will be a sum of spline functions each involving only one input variable. This is known as an additive function (no interactions)

$$\hat{f}(\mathbf{x}) = \sum_{j=1}^{J} f_j(x_j). \tag{13}$$

An additive function has the property that the functional dependence on any variable is independent of the values of all other input variables up to an overall additive constant. Additive function approximations are important because many true underlying functions $f(\mathbf{x})$ (2) are close to additive and thus well approximated by additive functions. MARS can realize additive functions as a subclass of its potential models.

Another potential network topology that can be realized by MARS is one in which every unit output serves either as an input to one (and only one) other unit or goes to the final weighted adder (but not both). This is a binary tree topology similar to those generated by recursive partitioning strategies like CART (Breiman, Friedman, Olshen and Stone [1984]). In fact, if one were to impose this restriction and employ $q = 0$ splines, the MARS strategy reduces to that of CART (see Friedman [1991]). Thus, MARS can also realize CART approximations as a subclass of its potential models.

MARS can be viewed as a generalization of CART. First by allowing $q > 0$ splines continuous approximations are produced. This generally results in a dramatic increase in accuracy. In addition, all unit outputs are eligible to contribute to the final adder, not just the terminal ones; and finally, all previous unit outputs are eligible to be selected as inputs for new units, not just the currently terminal ones.

Both additive and CART approximations have been highly successful in largely complementary situations: additive modeling when the true underlying function is close to additive, and CART when it dominately involves high order interactions between the input variables. MARS unifies both into a single framework. This lends hope that MARS will be successful at both these extremes as well as the broad spectrum of situations in between where neither works well.

Multiple response outputs $y_1 \cdots y_m$ (1) (2) are incorporated in a straightforward manner. The internal units and their interconnections are the same as described above and shown in Figures 2 and 3. Only the final weighted adder unit (Figure 2) is modified to incorporate a set of weights

$$\{w_{mk}\}_0^M \,_1^m \tag{14}$$

for each response output ($k = 1, m$). The approximation for each output is

$$\hat{f}_k(\mathbf{x}) = \sum_{m=0}^{M} w_{mk} B_m, \quad k = 1, m.$$

The numerator in the GCV criterion (12) is replaced by

$$\frac{1}{mN} \sum_{k=1}^{m} \sum_{i=1}^{N} (y_{ik} - \hat{f}_{ik})^2$$

and it is minimized with respect to the internal network parameters (10) and all of the weights (14).

## 4   DISCUSSION

This section (briefly) compares and contrasts the MARS approach with radial basis functions and sigmoid "back-probagation" networks. An important consequence of the MARS strategy is input variable subset selection. Each unit individually selects the best system input so that it can best contribute to the approximation. It is often the case that some or many of the inputs are never selected. These will be inputs that tend to have little or no effect on the output(s). In this case excluding them from the approximation will greatly increase statistical accuracy. It also aids in the interpretation of the produced model. In addition to global variable subset selection, MARS is able to do input variable subset selection locally in different regions of the input variable space. This is a consequence of the restricted support (nonzero value) of the basis functions produced. Thus, if in any local region, the target function (2) depends on only a few of the inputs, MARS is able to use this to advantage even if the relevant inputs are different in different local regions. Also, MARS is able to produce approximations of low interaction order even if the number of selected inputs is large.

Radial basis functions are not able to do local (or usually even global) input variable subset selection as a part of the procedure. All basis functions involve all of the inputs at the same relative strength everywhere in the input variable space. If the target function (2) is of this nature they will perform well in that no competing procedure will do better, or likely even as well. If this is not the case, radial basis functions are not able to take advantage of the situation to improve accuracy. Also, radially symmetric basis functions produce approximations of the highest possible interaction order (everywhere in the input space). This results in a marked disadvantage if the target function tends to dominately involve interactions in at most a few of the inputs (such as additive functions (13)).

Standard networks based on sigmoidal units of linear combinations of inputs share the properties described above for radial basis functions. Including "weight elimination" (Rumelhart [1988]) provides an (important) ability to do global (but not local) input variable subset selection. The principal differences between MARS and this approach center on the use of splines rather than sigmoids, and products rather than linear combinations of the input variables. Splines tend to be more flexible in that two spline functions can closely approximate any sigmoid whereas it can take many sigmoids to approximate some splines. MARS' use of product expansions enables it to produce approximations that are local in nature. Local approximations have the property that if the target function is badly behaved in any local region of the input space, the quality of the approximation is not affected in the other regions. Also, as noted above, MARS can produce approximations of low interaction order. This is difficult for approximations based on linear combinations.

Both radial basis functions and sigmoidal networks produce approximations that are difficult to interpret. Even in situations where they produce high accuracy, they provide little information concerning the nature of the target function. MARS approximations on the other hand can often provide considerable interpretable in-

formation. Interpreting MARS models is discussed in detail in Friedman [1991]. Finally, training MARS networks tends to be computationally much faster than other types of learning procedures.

## References

Bellman, R. E. (1961). *Adaptive Control Processes*. Princeton University Press, Princeton, NJ.

Breiman, L., Friedman, J. H., Olshen, R. A. and Stone, C. J. (1984). *Classification and Regression Trees*. Wadsworth, Belmont, CA.

Cheney, E. W. (1986). *Multivariate Approximation Theory: Selected Topics*. Monograph: SIAM CBMS-NSF Regional Conference Series in Applied Mathematics, Vol. 51.

Chui, C. K. (1988). *Multivariate Splines*. Monograph: SIAM CBMS-NSF Regional Conference Series in Applied Mathematics, Vol. 54.

Craven, P. and Wahba, G. (1979). Smoothing noisy data with spline functions. Estimating the correct degree of smoothing by the method of generalized cross-validation. *Numerische Mathematik* 31 317–403.

deBoor, C. (1978). *A Practical Guide to Splines*. Springer-Verlag, New York, NY.

Friedman, J. H. (1991). Multivariate adaptive regression splines (with discussion). *Annals of Statistics*, March.

Rumelhart, D. E. (1988). Learning and generalization. IEEE International Conference on Neural Networks, San Diego, plenary address.

FIGURE 1

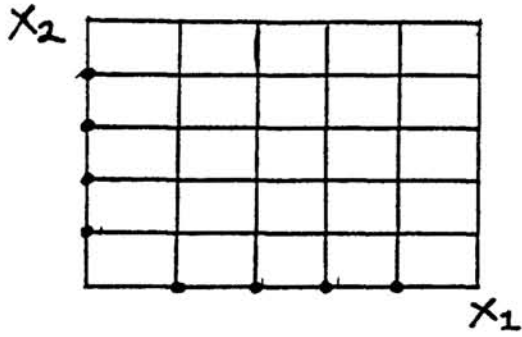

FIGURE 2
General Adaptive Spline Network

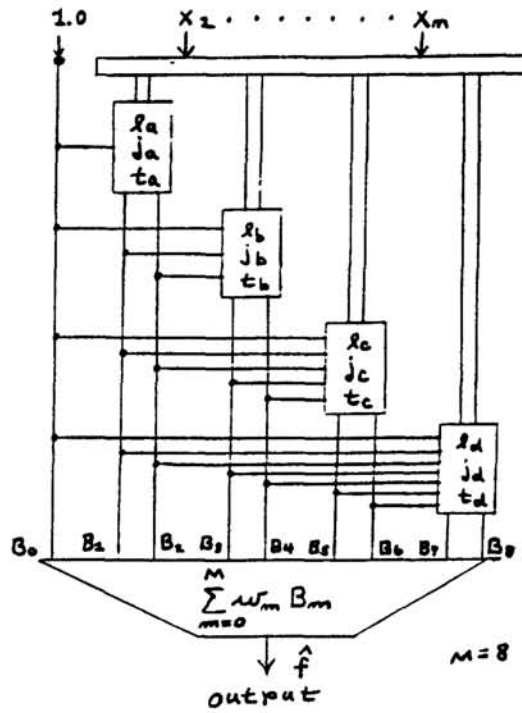

$$\sum_{m=0}^{M} w_m B_m$$

$\hat{f}$

output

$M = 8$

FIGURE 3
Adaptive Spline Unit

parameters: $\ell, j, t$

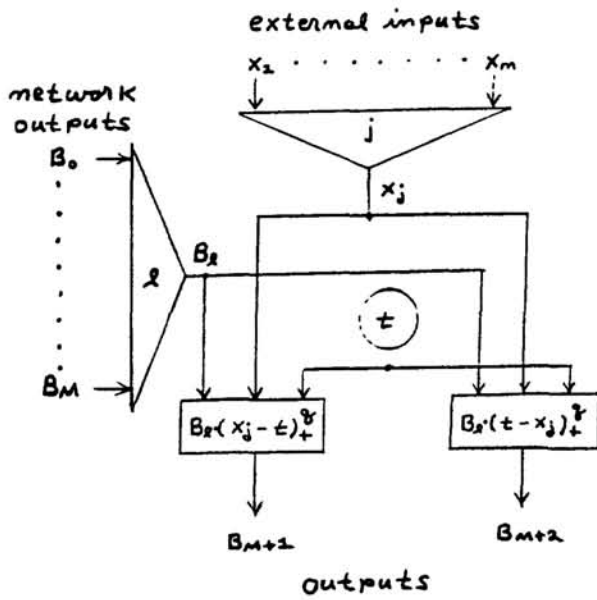

FIGURE 4

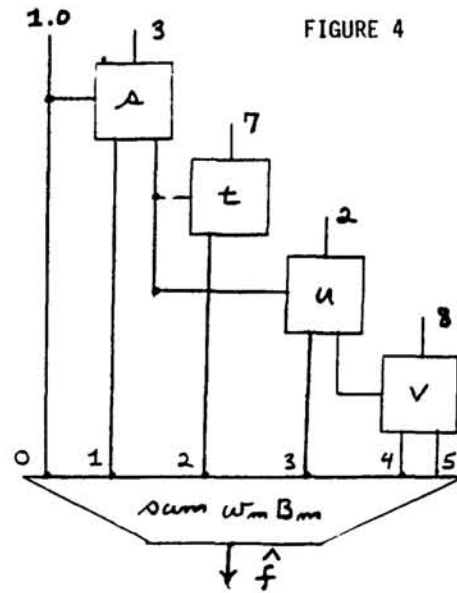

sum $w_m B_m$

$\hat{f}$